# Thin Junction Trees

**Francis R. Bach**
Computer Science Division
University of California
Berkeley, CA 94720
*fbach@cs.berkeley.edu*

**Michael I. Jordan**
Computer Science and Statistics
University of California
Berkeley, CA 94720
*jordan@cs.berkeley.edu*

## Abstract

We present an algorithm that induces a class of models with *thin junction trees*—models that are characterized by an upper bound on the size of the maximal cliques of their triangulated graph. By ensuring that the junction tree is thin, inference in our models remains tractable throughout the learning process. This allows both an efficient implementation of an iterative scaling parameter estimation algorithm and also ensures that inference can be performed efficiently with the final model. We illustrate the approach with applications in handwritten digit recognition and DNA splice site detection.

## Introduction

Many learning problems in complex domains such as bioinformatics, vision, and information retrieval involve large collections of interdependent variables, none of which has a privileged status as a response variable or class label. In such problems, the goal is generally that of characterizing the principal dependencies in the data, a problem which is often cast within the framework of multivariate density estimation. Simple models are often preferred in this setting, both for their computational tractability and their relative immunity to overfitting. Thus models involving low-order marginal or conditional probabilities—e.g., naive independence models, trees, or Markov models—are in wide use. In problems involving higher-order dependencies, however, such strong assumptions can be a serious liability.

A number of methods have been developed for selecting models of higher-order dependencies in data, either within the maximum entropy setting—in which features are selected [9, 16]—and the graphical model setting—in which edges are selected [8]. Simplicity also plays an important role in the design of these algorithms; in particular, greedy methods that add or subtract a single feature or edge at a time are generally employed. The model that results at each step of this process, however, is often not simple, and this is problematic both computationally and statistically in large-scale problems.

In the current paper we describe a methodology that can be viewed as a generalization of the Chow-Liu algorithm for constructing tree models [2]. Note that tree models have the property that their junction trees have no more than two nodes in any clique—the *treewidth* of tree models is one. In our generalization, we allow the treewidth to be a larger, but still controlled, value. We fit data within the space of models having "thin" junction trees.

Models with thin junction trees are tractable for exact inference, indeed the complexity of any type of inference (joint, marginal, conditional) is controlled by the upper bound that is imposed on the treewidth. This makes it possible to achieve some of the flexibility that is often viewed as a generic virtue of generative models, but is not always achievable in practice. For example, in the classification setting we are able to classify partially observed data (e.g., occluded digits) in a simple and direct way—we simply marginalize away the unobserved variables, an operation which is tractable in our models. We illustrate this capability in a study of handwritten digit recognition in Section 4.2, where we compare thin junction trees and support vector machines (SVMs), a discriminative technique which does not come equipped with a simple and principled method for handling partially observed data. As we will see, thin junction trees are quite robust to missing data in this domain.

There are a number of issues that need to be addressed in our framework. In particular, tree models come equipped with particularly efficient algorithms for parameter estimation and model selection—algorithms which do not generalize readily to non-tree models, including thin junction tree models. It is important to show that efficient algorithms can nonetheless be found to fit such models. We show how this can be achieved in Sections 1, 2 and 3. Empirical results using these algorithms are presented in Section 4.

# 1   Feature induction

We assume an input space $\mathcal{X}$ with $M$ variables and a target probability distribution $\tilde{p}$. Our goal is to find a probability distribution $q$ that minimizes the Kullback-Leibler divergence $D(\tilde{p} \parallel q)$. Consider a vector-valued "feature" or "sufficient statistic" $f : \mathcal{X} \to \mathbf{R}^F$, where $F$ is the dimensionality of the feature space. The feature $f$ can also be thought in terms of its components as a set of $F$ real-valued features $(f_i)$. We focus on exponential family distributions (also known as "Gibbs" or "maximum entropy" distributions) based on these features: $q(x) = q_0(x) \exp(\lambda \cdot f(x))/Z$ where $\lambda = (\lambda_i) \in \mathbf{R}^F$ is a parameter vector, $q_0$ is a base-measure (typically uniform), and $Z$ is the normalizing constant. (Section 3 considers the closely-related problem of inducing edges rather than features).

Each feature is a function of a certain subset of variables, and we let $T_k \subset \mathcal{V} = \{1, 2, \ldots, M\}$ index the subset of variables referred to by feature $f_k$. Let us consider the undirected graphical model $\mathcal{G} = (\mathcal{V}, \mathcal{E})$, where the set of edges $\mathcal{E}$ is the set of all pairs included in at least one $T_k$. With this definition the $T_k$ are the maximal cliques of the graph and, if $q_0$ is decomposable in this graph, the exponential family distribution with features $f$ and reference distribution $q_0$ is also decomposable in this graph. We assume without loss of generality that the graph is connected. For each possible triangulation of the graph, we can define a *junction tree* [4], where for all $k$ there exists a maximal clique containing $T_k$. The complexity of exact inference depends on the size of the maximal clique of the triangulated graph. We define the *treewidth* $\tau$ of our original graph to be one less than the minimum possible value of this maximal clique size for all possible triangulations. We say that a graphical model has a thin junction tree if its treewidth $\tau$ is small.

Our basic feature induction algorithm is a constrained variant of that proposed by [9]. Given a set of available features, we perform a greedy search to find the set of features that enables the best possible fit to $\tilde{p}$, under the constraint of having a thin junction tree. At each step, candidates are ranked according to the gain in KL divergence, with respect to the empirical distribution, that would be achieved by their addition to the current set of features. Features that would generate a graphical cover with treewidth greater than a given upper bound $\tau$ are removed from the ranking.

The parameter values $\lambda$ are held fixed during each step of the feature ranking process. Once a set of candidate features are chosen, however, we reestimate all of the parameters (using the algorithm to be described in Section 2) and iterate.

1. *Initialization*: $q = q_0$, $f = \varnothing$, $\lambda = \varnothing$, a set of available features
2. *Repeat* steps (a) to (d) until no further progress is made with respect to a model selection criterion (e.g., MDL or cross-validation)
   (a) *Ranking*: generate samples from $q$ and rank feature candidates according to the KL gain
   (b) *Elimination*: remove all candidates that would generate a model with treewidth greater than $\tau$
   (c) *Selection*: select the $m$ best features $g_1, \ldots, g_m$ and add them to $f$
   (d) *Parameter Estimation*: Estimate $\lambda$ using the junction tree implementation of Iterative Scaling (see Section 2)

Freezing the parameters during the feature ranking step is suboptimal, but it yields an essential computational efficiency. In particular, as shown by [9], under these conditions we can rank a new feature $f$ by solving a polynomial equation whose degree is the number of values $f$ can take minus one, and whose coefficients are expectations under $q$ of functions of $f$. This equation has only one root and can be solved efficiently by Newton's method. When the feature $f$ is binary the process is even more efficient—the equation is linear and can be solved directly. Consequently, with a single set of samples from $q$, we can rank many features very cheaply.

For the feature elimination operation, algorithms exist that determine in time linear in the number of nodes whether a graph has a treewidth smaller than $\tau$, and if so output a triangulation in which all cliques are of size less than $\tau$ [1]. These algorithms are super-exponential in $\tau$, however, and thus are applicable only to problems with small treewidths. In practice we have had success using fast heuristic triangulation methods [11] that allow us to guarantee the existence of a junction tree with a maximal clique no larger than $\tau$ for a given model. (This is a conservative technique that may occasionally throw out models that in fact have small treewidth).

A critical bottleneck in the algorithm is the parameter estimation step, and it is important to develop a parameter estimation algorithm that exploits the bounded treewidth property. We now turn to this problem.

## 2 Iterative Scaling using the junction tree

Fitting an exponential family distribution under expectation constraints is a well studied problem; the basic technique is known as *Iterative Scaling*. A generalization of Iterative Proportional Fitting (IPF), it updates the parameters $\lambda_i$ sequentially [5]. Algorithms that update the parameters in parallel have also been proposed; in particular the *Generalized Iterative Scaling* algorithm [6], which imposes the constraint that the features sum to one, and the *Improved Iterative Scaling* algorithm [9], which removes this constraint. These algorithms have an important advantage in our setting in that, for each set of parameter updates, they only require computations of expectation that can all be estimated with a single set of samples from the current distribution.

When the input dimensionality is large, however, we would like to avoid sampling algorithms altogether. To do so we exploit the bounded treewidth of our models. We present a novel algorithm that uses the junction tree and the structure of the problem to speed up parameter estimation. The algorithm generalizes to Gibbs distributions the "effective IPF" algorithm of [10].

When working with a junction tree, a efficient way of performing Iterative Scaling is to update parameters block by block so that each update is performed for a relatively small

number of features on a small number of variables. Each block can be fit with any parameter estimation algorithm, in particular Improved Iterative Scaling (IIS). The following algorithm exploits this idea by grouping the features whose supports are in the same clique of the triangulated graph. Thus, parameter estimation is done in spaces of dimensions at most $\tau + 1$, and all the needed expectations can be evaluated cheaply.

## 2.1 Notation

Let $f$ be our $F$-dimensional feature. Let $(C_i)_{0 \leqslant i \leqslant N_C}$ denote the maximal cliques of the triangulated graph, with potentials $\phi_{C_i}$. We assign each feature $f_k$ to one of the cliques $C_j$ that contains $T_k$. For each clique $C_j$ we denote $F_j = (f_{k_1}, \ldots, f_{k_{n_j}})$ as the set of features assigned to $C_j$.

## 2.2 Algorithm

EFFICIENTITERATIVESCALING

1. *Initialization*:
   –Construct a junction tree associated with the subsets $\{T_k = \operatorname{supp}(f_k)\}$
   –Assign each $f_k$ to one $C_j$, such that $T_k \subset C_j$ (equivalent to determining $F_j = (f_{k_1}, \ldots, f_{k_{n_j}})$ for all $j$)
   –Set $\lambda = (\lambda_1, \ldots, \lambda_F) = 0$ and decompose $q_0$ onto the junction tree
   –Set $q = q_0 = \prod_j \phi_{C_j}$

2. *Loop until convergence*: Repeat step (3) until convergence of the $\lambda$'s

3. *Loop through all cliques*: Repeat steps (a) to (c) for all cliques $C_j$

   (a) Define the root of the junction tree to be $C_j$
   (b) Collect evidence from the leaves to the root of the junction tree and normalize potential $\phi_{C_j}$
   (c) Calculate the maximum likelihood $|C_j|$-dimensional exponential family distribution with features $F_j$ and reference distribution $\phi_{C_j}$, using IIS. Replace $\phi_{C_j}$ by this distribution and add the resulting parameters (one for each feature in $F_j$) to the corresponding $\lambda$'s: $(\lambda_{k_1}, \ldots, \lambda_{k_{n_j}})$.

After step (b), the potential $\phi_{C_j}$ is exactly $q$ marginalized to $C_j$, so that performing IIS for the features $F_j$ can be done using $\phi_{C_j}$ instead of the full distribution $q$. Moreover, each pass through all the cliques is equivalent to one pass of Iterative Scaling and therefore this algorithm converges to the maximum likelihood distribution.

## 3 Edge induction

Thus far we have emphasized the exponential family representation. Our algorithm can, however, be adapted readily to the problem of learning the structure of a graphical model. This is achieved by using features that are indicators of subsets of variables, ensuring that there is one such indicator for every combination of values of the variables in a clique. In this case, Iterative Scaling reduces to Iterative Proportional Fitting.

We generally employ a further approximation when ranking and selecting edges. In particular, we evaluate an edge only in terms of the two variables associated directly with the edge. The clique formed by the addition of the edge, however, may involve additional higher-order dependencies, which can be parameterized and incorporated in the model. Evaluating edges in this way thus underestimates the potential gain in KL divergence.

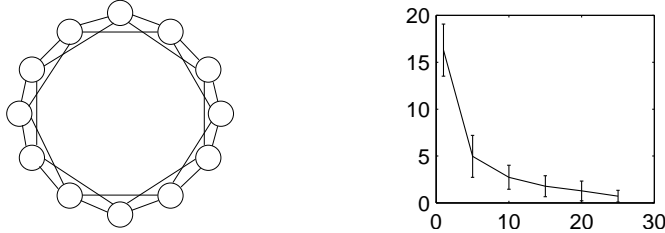

Figure 1: (Left) Circular Boltzmann machine of treewidth 4. (Right) Proportion (in %) of edges not in common between the fitted model and the generating model vs the number of available training examples (in thousands).

We should not expect to be able to find an exact edge-selection method—recent work by Srebro [15] has shown that the related problem of finding the maximum likelihood graphical model with bounded treewidth is NP-hard.

## 4 Empirical results

### 4.1 Small graphs with known generative model

In this experiment we generate samples from a known graphical model and fit our model to the data. We consider circular Boltzmann machines of known treewidth equal to 4 as shown in Figure 1. Our networks all have 32 nodes and the weights were selected from a uniform distribution in $[-2; -1] \cup [1; 2]$—so that each edge is significant. For an increasing number of training samples, ten replications were performed for each case using our feature induction algorithm with maximum treewidth equal to 4. Figure 1 shows that with enough samples we are able to recover the structure almost exactly (up to $0.7\%$ of the original edges).

### 4.2 MNIST digit dataset

In this section we study the performance of the thin junction tree method on the MNIST dataset of handwritten digits. While discriminative methods outperform generative methods in this high-dimensional setting [12], generative methods offer capabilities that are not provided by discriminative classifiers; in particular, the ability to deal with large fractions of missing pixels and the ability to to reconstruct images from partial data. It is of interest to see how much performance loss we incur and how much robustness we gain by using a sophisticated generative model for this problem.

The MNIST training set is composed of $28 \times 28$ 4-bit grayscale pixels that have been resized and cropped to $16 \times 16$ binary images (an example is provided in the leftmost plots in Figure 2). We used thin junction trees as density estimators in the 256-dimensional pixel space by training ten different models, one for each of the ten classes. We used binary features of the form $\prod_j \delta(x_{i_j} = 1)$. No vision-based techniques such as de-skewing or virtual examples were used. We utilized ten percent fractions of the training data for cross-validation and test.

*Density estimation*: The leftmost plot in Figure 3 shows how increasing the maximal allowed treewidth, ranging from 1 (trees) to 15, enables a better fit to data.

*Classification*: We built classifiers from the bank of ten thin junction tree ("*TJT*") models using one of the following strategies: (1) take the maximum likelihood among the ten

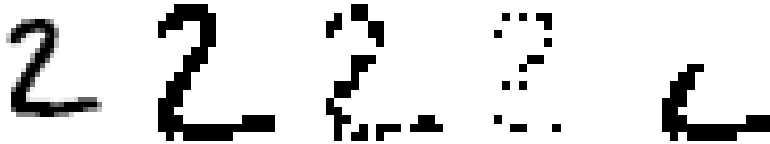

Figure 2: Digit from the MNIST database. From left to right, original digit, cropped and resized digits used in our experiments, 50% of missing values, 75% of missing values, occluded digit.

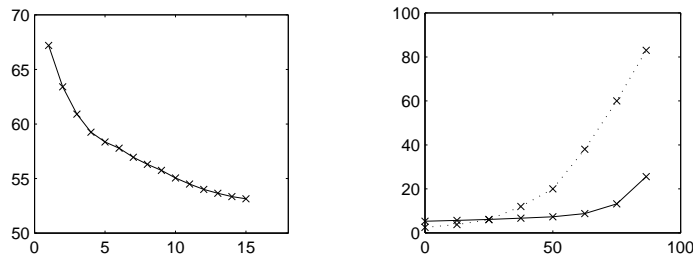

Figure 3: (Left) Negative log likelihood for the digit 2 vs maximal allowed treewidth. (Right) Error rate as a function of the percentage of erased pixels for the TJT classifier (plain) and a support vector machine (dotted). See text for details.

models (*TJT-ML*), or (2) train a discriminative model using the outputs of the ten models. We used softmax regression (*TJT-Softmax*) and the support vector machine (*TJT-SVM*) in the latter case.

The classification error rates were as follows: *LeNet* 0.7, *SVM* 0.8, *Product of experts*, 2.0, *TJT-SVM* 3.8, *TJT-Softmax* 4.2, *TJT-ML* 5.3, *Chow-Liu* 8.5, and *Linear classifier* 12.0. (See [12] and [13] for further details on the non-TJT models).

It is important to emphasize that our models are tractable for full joint inference; indeed, the junction trees have a maximal clique size of 10 in the largest models we used on the ten classes. Thus we can use efficient exact calculations to perform inference. The following two sections demonstrate the utility of this fact.

*Missing pixels*: We ran an experiment in which pixels were chosen uniformly at random and erased, as shown in Figure 2. In our generative model, we treat them as hidden variables that were marginalized out. The rightmost plot in Figure 3 shows the error rate on the testing set as a function of the percentage of unknown pixels, for our models and for a SVM. In the case of the SVM, we used a polynomial kernel of degree four [7] and we tried various heuristics to fill in the value of the non-observed pixels, such as the average of that pixel over the training set or the value of a blank pixel. Best classification performance was achieved with replacing the missing value by the value of a blank pixel. Note that very little performance decrement is seen for our classifier even with up to 50 percent of the pixels missing, while for the SVM, although performance is better for small percentages, performance degrades more rapidly as the percentage of erased digits increases.

*Reconstruction*: We conducted an additional experiment in which the upper halves of images were erased. We ran the junction tree inference algorithm to fill in these missing values, choosing the maximizing value of the conditional probability (max-propagation). Figure 3 shows the results. For each line, from left to right, we show the original digit, the digit after erasure, reconstructions based on the model having the maximum likelihood, and

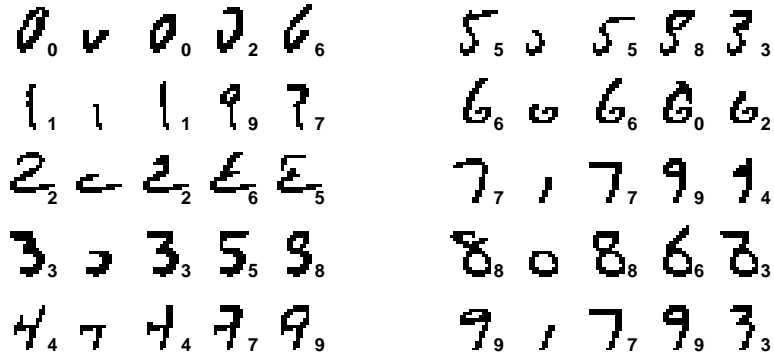

Figure 4: Reconstructions of images whose upper halves have been deleted. See text for details.

reconstruction based on the model having the second and third largest values of likelihood.

### 4.3 SPLICE Dataset

The task in this dataset is to classify splice junctions in DNA sequences. Splice junctions can either be an exon/intron (EI) boundary, an intron/exon (IE) boundary, or no boundary. (Introns are the portions of genes that are spliced out during transcription; exons are retained in the mRNA).

Each sample is a sequence of 60 DNA bases (where each base can take one of four values, A,G,C, or T). The three different classes are: EI exactly at the middle (between the 30th and the 31st bases), IE exactly at the middle (between the 30th and the 31st bases), no splice junction. The dataset is composed of 3175 training samples. In order to be able to compare to previous experiments using this dataset, performance is assessed by picking 2000 training data points at random and testing on the 1175 others, with 20 replications.

We treat classification as a density estimation problem in this case by treating the class variable $y$ as another variable. We classify by choosing the value of $y$ that maximizes the conditional probability $p(y|x)$. We tested both feature induction and edge induction; in the former case only binary features that are products of features of the form $\delta(x_i = a_i)$ were tested and induced. MDL was used to pick the number of features or edges.

Our feature induction algorithm, with a maximum treewidth equal to 5, gave an error rate of 3.4%, while the edge induction algorithm gave an error rate of 4.1%. This is better than the best reported results in the literature; in particular, neural networks have an error rate of 5.5% and the Chow and Liu algorithm has an error rate of 4.4% [14].

## 5  Conclusions

We have described a methodology for feature selection, edge selection and parameter estimation that can be viewed as a generalization of the Chow-Liu algorithm. Drawing on the feature selection methods of [9, 16], our method is quite general, building an exponential family model from the general vocabulary of features on overlapping subsets of variables. By maintaining tractability throughout the learning process, however, we build this flexible representation of a multivariate density while retaining many of the desirable aspects of the Chow-Liu algorithm.

Our methodology applies equally well to feature or edge selection. In large-scale, sparse domains in which overfitting is of particular concern, however, feature selection may be the preferred approach, in that it provides a finer-grained search in the space of simple models than is allowed by the edge selection approach.

## Acknowledgements

We wish to acknowledge NSF grant IIS-9988642 and ONR MURI N00014-00-1-0637. The results presented here were obtained using Kevin Murphy's Bayes Net Matlab toolbox and SVMTorch [3].

## References

[1] H. Bodlaender, A linear-time algorithm for finding tree-decompositions of small treewidth, *Siam J. Computing*, *25*, 105-1317, 1996.

[2] C.K. Chow and C.N. Liu, Approximating discrete probability distributions with dependence trees, *IEEE Trans. Information Theory*, *42*, 393-405, 1990.

[3] R. Collobert and S. Bengio, SVMTorch: support vector machines for large-scale regression problems, *Journal of Machine Learning Research*, *1*, 143-160, 2001.

[4] R.G. Cowell, A.P. Dawid, S.L. Lauritzen, and D.J. Spiegelhalter, *Probabilistic Networks and Expert Systems*, Springer-Verlag, New York, 1999.

[5] I. Csiszár, I-divergence geometry of probability distributions and minimization problems, *Annals of Probability*, *3*, 146-158, 1975.

[6] J.N. Darroch and D. Ratcliff, Generalized iterative scaling for log-linear models, *Ann. Math. Statist.*, *43*, 1470-1480, 1972.

[7] D. DeCoste and B. Schölkopf, Training invariant support vector machines, *Machine Learning*, *46*, 1-3, 2002.

[8] D. Heckerman, D. Geiger, and D.M. Chickering, Learning Bayesian networks: The combination of knowledge and statistical data, *Machine Learning*, *20*, 197-243, 1995.

[9] S. Della Pietra, V. Della Pietra, and J. Lafferty, Inducing features of random fields, *IEEE Trans. PAMI*, *19*, 380-393, 1997.

[10] R. Jirousek and S. Preucil, On the effective implementation of the iterative proportional fitting procedure, *Computational Statistics and Data Analysis*, *19*, 177-189, 1995.

[11] U. Kjaerulff, *Triangulation of graphs—algorithms giving small total state space*, Technical Report R90-09, Dept. of Math. and Comp. Sci., Aalborg Univ., Denmark, 1990.

[12] Y. Le Cun, `http://www.research.att.com/~yann/exdb/mnist/index.html`

[13] G. Mayraz and G. Hinton, Recognizing hand-written digits using hierarchical products of experts, *Adv. NIPS* 13, MIT Press, Cambridge, MA, 2001.

[14] M. Meila and M.I. Jordan, Learning with mixtures of trees, *Journal of Machine Learning Research*, *1*, 1-48, 2000.

[15] N. Srebro, Maximum likelihood bounded tree-width Markov networks, in *UAI 2001*.

[16] S.C. Zhu, Y.W. Wu, and D. Mumford, Minimax entropy principle and its application to texture modeling, *Neural Computation*, *9*, 1997.
